# Submanifold density estimation

**Arkadas Ozakin**
Georgia Tech Research Institute
Georgia Insitute of Technology
arkadas.ozakin@gtri.gatech.edu

**Alexander Gray**
College of Computing
Georgia Institute of Technology
agray@cc.gatech.edu

## Abstract

Kernel density estimation is the most widely-used practical method for accurate nonparametric density estimation. However, long-standing worst-case theoretical results showing that its performance worsens exponentially with the dimension of the data have quashed its application to modern high-dimensional datasets for decades. In practice, it has been recognized that often such data have a much lower-dimensional intrinsic structure. We propose a small modification to kernel density estimation for estimating probability density functions on Riemannian submanifolds of Euclidean space. Using ideas from Riemannian geometry, we prove the consistency of this modified estimator and show that the convergence rate is determined by the intrinsic dimension of the submanifold. We conclude with empirical results demonstrating the behavior predicted by our theory.

## 1 Introduction: Density estimation and the curse of dimensionality

Kernel density estimation (KDE) [8] is one of the most popular methods for estimating the underlying probability density function (PDF) of a dataset. Roughly speaking, KDE consists of having the data points "contribute" to the estimate at a given point according to their distances from the point. In the simplest multi-dimensional KDE [3], the estimate $\hat{f}_m(\mathbf{y}_0)$ of the PDF $f(\mathbf{y}_0)$ at a point $\mathbf{y}_0 \in \mathbb{R}^N$ is given in terms of a sample $\{\mathbf{y}_1, \ldots, \mathbf{y}_m\}$ as,

$$\hat{f}_m(\mathbf{y}_0) = \frac{1}{m} \sum_{i=1}^{m} \frac{1}{h_m^N} K\left(\frac{\|\mathbf{y}_i - \mathbf{y}_0\|}{h_m}\right), \tag{1}$$

where $h_m > 0$, the *bandwidth*, is chosen to approach to zero at a suitable rate as the number $m$ of data points increases, and $K : [0.\infty) \rightarrow [0, \infty)$ is a *kernel function* that satisfies certain properties such as boundedness. Various theorems exist on the different types of convergence of the estimator to the correct result and the rates of convergence. The earliest result on the pointwise convergence rate in the multivariable case seems to be given in [3], where it is stated that under certain conditions for $f$ and $K$, assuming $h_m \rightarrow 0$ and $mh_m \rightarrow \infty$ as $m \rightarrow \infty$, the mean squared error in the estimate $\hat{f}(\mathbf{y}_0)$ of the density at a point goes to zero with the rate, $\text{MSE}[\hat{f}_m(\mathbf{y}_0)] = \text{E}\left[\left(\hat{f}_m(\mathbf{y}_0) - f(\mathbf{y}_0)\right)^2\right] = O\left(h_m^4 + \frac{1}{mh_m^N}\right)$ as $m \rightarrow \infty$. If $h_m$ is chosen to be proportional to $m^{-1/(N+4)}$, one gets,

$$\text{MSE}[\hat{f}_m(p)] = O\left(\frac{1}{m^{4/(N+4)}}\right), \tag{2}$$

as $m \rightarrow \infty$. This is an example of a *curse of dimensionality*; the convergence rate slows as the dimensionality $N$ of the data set increases. In Table 4.2 of [12], Silverman demonstrates how the sample size required for a given mean square error for the estimate of a multivariable normal distribution increases with the dimensionality. The numbers look as discouraging as the formula 2.

One source of optimism towards various curses of dimensionality is the fact that although the data for a given problem may have many features, in reality the *intrinsic* dimensionality of the "data subspace" of the full feature space may be low. This may result in there being no curse at all, if the performance of the method/algorithm under consideration can be shown to depend only on the intrinsic dimensionality of the data. Alternatively, one may be able to avoid the curse by devising ways to work with the low-dimensional data subspace by using dimensional reduction techniques on the data. One example of the former case is the results on nearest neighbor search [6, 2] which indicate that the performance of certain nearest-neighbor search algortihns is determined not by the full dimensionality of the feature space, but only on the intrinsic dimensionality of the data subspace.

**Riemannian manifolds.** In this paper, we will assume that the data subspace is a Riemannian manifold. Riemannian manifolds provide a generalization of the notion of a smooth surface in $\mathbb{R}^3$ to higher dimensions. As first clarified by Gauss in the two-dimensional case (and by Riemann in the general case) it turns out that *intrinsic* features of the geometry of a surface such as lengths of its curves or intrinsic distances between its points, etc., can be given in terms of the so-called metric tensor[1] $\mathbf{g}$ without referring to the particular way the the surface is embedded in $\mathbb{R}^3$. A space whose geometry is defined in terms of a metric tensor is called a Riemannian manifold (for a rigorous definition, see, e.g., [5, 7, 1]).

**Previous work.** In [9], Pelletier defines an estimator of a PDF on a Riemannian manifold $M$ by using the distances measured on $M$ via its metric tensor, and obtains the same convergence rate as in (2), with $N$ being replaced by the dimensionality of the Riemannian manifold. Thus, if we know that the data lives on a Riemannian manifold $M$, the convergence rate of this estimator will be determined by the dimensionality of $M$, instead of the full dimensionality of the feature space on which the data may have been originally sampled. While an interesting generalization of the usual KDE, this approach assumes that the data manifold $M$ is known in advance, and that we have access to certain geometric quantities related to this manifold such as intrinsic distances between its points and the so-called *volume density function*. Thus, this Riemannian KDE cannot be used directly in a case where the data lives on an *unknown* Riemannian submanifold of $\mathbb{R}^N$. Certain tools from existing nonlinear dimensionality reduction methods could perhaps be utilized to estimate the quantities needed in the estimator of [9], however, a more straightforward method that directly estimates the density of the data as measured in the subspace is desirable.

Other related works include [13], where the authors propose a submanifold density estimation method that uses a kernel function with a variable covariance but do not present theorerical results, [4] where the author proposes a method for doing density estimation on a Riemannian manifold by using the eigenfunctions of the Laplace-Beltrami operator, which, as in [9], assumes that the manifold is known in advance, together with intricate geometric information pertaining to it, and [10, 11], which discuss various issues related to statistics on a Riemannian manifold.

**This paper.** In this paper, we propose a direct way to estimate the density of Euclidean data that lives on a Riemannian submanifold of $\mathbb{R}^N$ with *known dimension* $n < N$. We prove the pointwise consistency of the estimator, and prove bounds on its convergence rates given in terms of the intrinsic dimension of the submanifold the data lives in. This is an example of the avoidance of the curse of dimensionality in the manner mentioned above, by a method whose performance depends on the intrinsic dimensionality of the data instead of the full dimensionality of the feature space. Our method is practical in that it works with Euclidean distances on $\mathbb{R}^N$. In particular, we do not assume any knowledge of the quantities pertaining to the intrinsic geometry of the underlying submanifold such as its metric tensor, geodesic distances between its points, its volume form, etc.

## 2 The estimator and its convergence rate

**Motivation.** In this paper, we are concerned with the estimation of a PDF that lives on an (unknown) $n$-dimensional Riemannian submanifold $M$ of $\mathbb{R}^N$, where $N > n$. Usual, $N$-dimensional kernel density estimation would not work for this problem, since if interpreted as living on $\mathbb{R}^N$, the

underlying PDF would involve a "delta function" that vanishes when one moves away from $M$, and "becomes infinite" on $M$ in order to have proper normalization. More formally, the $N$-dimensional probability measure for such an $n$-dimensional PDF on $M$ will have support only on $M$, will not be absolutely continuous with respect to the Lebesgue measure on $\mathbb{R}^N$, and will not have a probability density function on $\mathbb{R}^N$. If one attempts to use the usual, $N$-dimensional KDE for data drawn from such a probability measure, the estimator will "try to converge" to a singular PDF, one that is infinite on $M$, zero outside.

In order to estimate the probability density function on $M$ by using data given in $\mathbb{R}^N$, we propose a simple modification of usual KDE on $\mathbb{R}^N$, namely, to use a kernel that is normalized for $n$-dimensions instead of $N$, while still using the Euclidean distances in $\mathbb{R}^N$. The intuition behind this approach is based on three facts: 1) For small distances, an $n$-dimensional Riemannian manifold "looks like" $\mathbb{R}^n$, and densities in $\mathbb{R}^n$ should be estimated by an $n$-dimensional kernel, 2) For points of $M$ that are close enough to each other, the intrinsic distances as measured on $M$ are close to Euclidean distances as measured in $\mathbb{R}^N$, and, 3) For small bandwidths, the main contribution to the estimate at a point comes from data points that are nearby. Thus, as the number of data points increases and the bandwidth is taken to be smaller and smaller, estimating the density by using a kernel normalized for $n$-dimensions and distances as measured in $\mathbb{R}^N$ should give a result closer and closer to the correct value.

We will next give the formal definition of the estimator motivated by these considerations, and state our theorem on its asymptotics. As in the original work of Parzen [8], the proof that the estimator is asymptotically unbiased consists of proving that as the bandwidth converges to zero, the kernel function becomes a "delta function". This result is also used in showing that with an appropriate choice of vanishing rate for the bandwidth, the variance also vanishes asymptotically, hence the estimator is pointwise consistent.

**Statement of the theorem**   Let $M$ be an $n$-dimensional, embedded, complete Riemannian submanifold of $\mathbb{R}^N$ ($n < N$) with an induced metric $\mathbf{g}$ and injectivity radius $r_{inj} > 0$.[2] Let $d(p,q)$ be the length of a length-minimizing geodesic in $M$ between $p, q \in M$, and let $u(p,q)$ be the geodesic (linear) distance between $p$ and $q$ as measured in $\mathbb{R}^N$. Note that $u(p,q) \leq d(p,q)$. We will use the notation $u_p(q) = u(p,q)$ and $d_p(q) = d(p,q)$. We will denote the Riemannian volume measure on $M$ by $V$, and the volume form by $dV$.

**Theorem 2.1.** *Let $f : M \to [0, \infty)$ be a probability density function defined on $M$ (so that the related probability measure is $fV$), and $K : [0, \infty) \to [0, \infty)$ be a continous function that satisfies vanishes outside $[0, 1)$, is differentiable with a bounded derivative in $[0, 1)$, and satisfies, $\int_{\|\mathbf{z}\| \leq 1} K(\|\mathbf{z}\|) d^n \mathbf{z} = 1$. Assume $f$ is differentiable to second order in a neighborhood of $p \in M$, and for a sample $q_1, \ldots, q_m$ of size $m$ drawn from the density $f$, define an estimator $\hat{f}_m(p)$ of $f(p)$ as,*

$$\hat{f}_m(p) = \frac{1}{m} \sum_{j=1}^{m} \frac{1}{h_m^n} K\left( \frac{u_p(q_j)}{h_m} \right) \tag{3}$$

*where $h_m > 0$. If $h_m$ satisfies $\lim_{m \to \infty} h_m = 0$ and $\lim_{m \to \infty} m h_m^n = \infty$, then, there exists non-negative numbers $m_*$, $C_b$, and $C_V$ such that for all $m > m_*$ we have,*

$$\text{MSE}\left[ \hat{f}_m(p) \right] = \text{E}\left[ \left( \hat{f}_m(p) - f(p) \right)^2 \right] < C_b h_m^4 + \frac{C_V}{m h_m^n} . \tag{4}$$

*If $h_m$ is chosen to be proportional to $m^{-1/(n+4)}$, this gives, $\text{E}\left[ (f_m(p) - f(p))^2 \right] = O\left( \frac{1}{m^{4/(n+4)}} \right)$ as $m \to \infty$.*

Thus, the convergence rate of the estimator is given as in [3, 9], with the dimensionality replaced by the intrinsic dimension $n$ of $M$. The proof will follow from the two lemmas below on the convergence rates of the bias and the variance.

## 3 Preliminary results

The following theorem, which is analogous to Theorem 1A in [8], tells that up to a constant, the kernel becomes a "delta function" as the bandwith gets smaller.

**Theorem 3.1.** *Let $K : [0, \infty) \to [0, \infty)$ be a continuous function that vanishes outside $[0, 1)$ and is differentiable with a bounded derivative in $[0, 1)$, and let $\xi : M \to \mathbb{R}$ be a function that is differentiable to second order in a neighborhood of $p \in M$. Let*

$$\xi_h(p) = \frac{1}{h^n} \int_M K \left( \frac{u_p(q)}{h} \right) \xi(q) \, dV(q) \,, \tag{5}$$

*where $h > 0$ and $dV(q)$ denotes the Riemannian volume form on $M$ at point $q$. Then, as $h \to 0$,*

$$\xi_h(p) - \xi(p) \int_{\mathbb{R}^n} K(\|\mathbf{z}\|) d^n \mathbf{z} = O(h^2) \,, \tag{6}$$

*where $\mathbf{z} = (z^1, \dots, z^n)$ denotes the Cartesian coordinates on $\mathbb{R}^n$ and $d^n \mathbf{z} = dz^1 \dots dz^n$ denotes the volume form on $\mathbb{R}^n$. In particular, $\lim_{h \to 0} \xi_h(p) = \xi(p) \int_{\mathbb{R}^n} K(\|\mathbf{z}\|) d^n \mathbf{z}$.*

Before proving this theorem, we prove some results on the relation between $u_p(q)$ and $d_p(q)$.

**Lemma 3.1.** *There exist $\delta_{u_p} > 0$ and $M_{u_p} > 0$ such that for all $q$ with $d_p(q) \leq \delta_{u_p}$, we have,*

$$d_p(q) \geq u_p(q) \geq d_p(q) - M_{u_p} \left[ d_p(q) \right]^3 \,. \tag{7}$$

*In particular, $\lim_{q \to p} \frac{u_p(q)}{d_p(q)} = 1$.*

*Proof.* Let $c_{\mathbf{v}_0}(s)$ be a geodesic in $M$ parametrized by arclength $s$, with $c(0) = p$ and initial velocity $\frac{dc_{\mathbf{v}_0}}{ds} \big|_{s=0} = \mathbf{v}_0$. When $s < r_{inj}$, $s$ is equal to $d_p(c_{\mathbf{v}_0}(s))$ [7, 1]. Now let $\mathbf{x}_{\mathbf{v}_0}(s)$ be the representation of $c_{\mathbf{v}_0}(s)$ in $\mathbb{R}^N$ in terms of Cartesian coordinates with the origin at $p$. We have $u_p(c_{\mathbf{v}_0}(s)) = \|\mathbf{x}_{\mathbf{v}_0}(s)\|$ and $\|\mathbf{x}'_{\mathbf{v}_0}(s)\| = 1$, which gives[3] $\mathbf{x}'_{\mathbf{v}_0}(s) \cdot \mathbf{x}''_{\mathbf{v}_0}(s) = 0$. Using these we get, $\frac{du_p(c_{\mathbf{v}_0}(s))}{ds} \big|_{s=0} = 1$, and $\frac{d^2 u_p(c_{\mathbf{v}_0}(s))}{ds^2} \big|_{s=0} = 0$. Let $M_3 \geq 0$ be an upper bound on the absolute value of the third derivative of $u_p(c_{\mathbf{v}_0}(s))$ for all $s \leq r_{inj}$ and all unit length $\mathbf{v}_0$: $\left| \frac{d^3 u_p(c_{\mathbf{v}_0}(s))}{ds^3} \right| \leq M_3$. Taylor's theorem gives $u_p(c_{\mathbf{v}_0}(s)) = s + R_{\mathbf{v}_0}(s)$ where $|R_{\mathbf{v}_0}(s)| \leq M_3 \frac{s^3}{3!}$. Thus, (7) holds with $M_{u_p} = \frac{M_3}{3!}$, for all $r < r_{inj}$. For later convenience, instead of $\delta_u = r_{inj}$, we will pick $\delta_{u_p}$ as follows. The polynomial $r - M_{u_p} r^3$ is monotonically increasing in the interval $0 \leq r \leq 1/\sqrt{3M_{u_p}}$. We let $\delta_{u_p} = \min\{r_{inj}, 1/\sqrt{M_{u_p}}\}$, so that $r - M_{u_p} r^3$ is ensured to be monotonic for $0 \leq r \leq \delta_{u_p}$. $\qquad\square$

**Definition 3.2.** *For $0 \leq r_1 < r_2$, let,*

$$\begin{aligned} H_p(r_1, r_2) &= \inf\{u_p(q) : r_1 \leq d_p(q) < r_2\} \,, &\tag{8}\\ H_p(r) &= H_p(r, \infty) = \inf\{u_p(q) : r_1 \leq d_p(q)\} \,, &\tag{9} \end{aligned}$$

*i.e., $H_p(r_1, r_2)$ is the smallest $u$-distance from $p$ among all points that have a $d$-distance between $r_1$ and $r_2$.*

Since $M$ is assumed to be an embedded submanifold, we have $H_p(r) > 0$ for all $r > 0$. In the below, we will assume that all radii are smaller than $r_{inj}$, in particular, a set of the form $\{q : r_1 \leq d_p(q) < r_2\}$ will be assumed to be non-empty and so, due to the completeness of $M$, to contain a point $q \in M$ such that $d_p(q) = r_1$. Note that,

$$H_p(r_1) = \min\{H(r_1, r_2), H(r_2)\} \,. \tag{10}$$

**Lemma 3.2.** *$H_p(r)$ is a non-decreasing, non-negative function, and there exist $\delta_{H_p} > 0$ and $M_{H_p} \geq 0$ such that, $r \geq H_p(r) \geq r - M_{H_p} r^3$, for all $r < \delta_{H_p}$. In particular, $\lim_{r \to 0} \frac{H_p(r)}{r} = 1$.*

*Proof.* $H_p(r)$ is clearly non-decreasing and $H_p(r) \leq r$ follows from $u_p(q) \leq d_p(q)$ and the fact that there exists at least one point $q$ with $d_p(q) = r$ in the set $\{q : r \leq d_p(q)\}$

Let $\delta_{H_p} = H_p(\delta_{u_p})$ where $\delta_{u_p}$ is as in the proof of Lemma 3.1 and let $r < \delta_{H_p}$. Since $r < \delta_{H_p} = H_p(\delta_{u_p}) \leq \delta_{u_p}$, by Lemma 3.1 we have,

$$r \geq u_p(r) \geq r - M_{u_p}r^3 , \tag{11}$$

for some $M_{u_p} > 0$. Now, since $r$ and $r - M_{u_p}r^3$ are both monotonic for $0 \leq r \leq \delta_{u_p}$, we have (see figure)

$$r \geq H_p(r, \delta_{u_p}) \geq r - M_{u_p}r^3 . \tag{12}$$

In particular, $H(r, \delta_{u_p}) \leq r < \delta_{H_p} = H_p(\delta_{u_p})$, i.e, $H(r, \delta_{u_p}) < H_p(\delta_{u_p})$. Using (10) this gives, $H_p(r) = H_p(r, \delta_{u_p})$. Combining this with (12), we get $r \geq H_p(r) \geq r - M_{u_p}r^3$ for all $r < \delta_{H_p}$. $\qquad\square$

Next we show that for all small enough $h$, there exists some radius $R_p(h)$ such that for all points $q$ with a $d_p(q) \geq R_p(h)$, we have $u_p(q) \geq h$. $R_p(h)$ will roughly be the inverse function of $H_p(r)$.

**Lemma 3.3.** *For any $h < H_p(r_{inj})$, let $R_p(h) = \sup\{r : H_p(r) \leq h\}$. Then, $u_p(q) \geq h$ for all $q$ with $d_p(q) \geq R_p(h)$ and there exist $\delta_{R_p} > 0$ and $M_{R_p} > 0$ such that for all $h \leq \delta_{R_p}$, $R_p(h)$ satisfies,*

$$h \leq R_p(h) \leq h + M_{R_p}h^3 . \tag{13}$$

*In particular,* $\lim_{h \to 0} \frac{R_p(h)}{h} = 1$.

*Proof.* That $u_p(q) \geq h$ when $d_q(q) \geq R_p(h)$ follows from the definitions. In order to show (13), we will use Lemma 3.2. Let $\alpha(r) = r - M_{H_p}r^3$, where $M_{H_p}$ is as in Lemma 3.2. Then, $\alpha(r)$ is one-to-one and continuous in the interval $0 \leq r \leq \delta_{H_p} \leq \delta_{u_p}$. Let $\beta = \alpha^{-1}$ be the inverse function of $\alpha$ in this interval. From the definition of $R_p(h)$ and Lemma 3.2, it follows that $h \leq R_p(h) \leq \beta(h)$ for all $h \leq \alpha(\delta_{H_p})$. Now, $\beta(0) = 0$, $\beta'(0) = 1$, $\beta''(0) = 0$, so by Taylor's theorem and the fact that the third derivative of $\beta$ is bounded in a neighborhood of 0, there exists $\delta_g$ and $M_{R_p}$ such that $\beta(h) \leq h + M_{R_p}h^3$ for all $h \leq \delta_g$. Thus,

$$h \leq R_p(h) \leq h + M_{R_p}h^3, \tag{14}$$

for all $h \leq \delta_R$ where $\delta_R = \min\{\alpha(\delta_{H_p}), \delta_g\}$. $\qquad\square$

**Proof of Theorem 3.1.** We will begin by proving that for small enough $h$, there is no contribution to the integral in the definition of $\xi_h(p)$ (see (5)) from outside the coordinate patch covered by normal coordinates.[4]

Let $h_0 > 0$ be such that $R_p(h_0) < r_{inj}$ (such an $h_0$ exists since $\lim_{h \to 0} R_p(h) = 0$). For any $h \leq h_0$, all points $q$ with $d_p(q) > r_{inj}$ will satisfy $u_p(q) > h$. This means if $h$ is small enough, $K(\frac{u_p(q)}{h}) = 0$ for all points outside the injectivity radius and we can perform the integral in (5) solely in the patch of normal coordinates at $p$.

For normal coordinates $\mathbf{y} = (y^1, \ldots, y^n)$ around the point $p$ with $\mathbf{y}(p) = \mathbf{0}$, we have $d_p(q) = \|\mathbf{y}(q)\|$ [7, 1]. With slight abuse of notation, we will write $u_p(\mathbf{y}(q)) = u_p(q)$, $\xi(\mathbf{y}(q)) = \xi(q)$ and $\mathbf{g}(q) = \mathbf{g}(\mathbf{y}(q))$, where $\mathbf{g}$ is the metric tensor of $M$.

Since $K(\frac{u_p(q)}{h}) = 0$ for all $q$ with $d_p(q) > R_p(h)$, we have,

$$\xi_h(p) = \frac{1}{h^n} \int_{\|\mathbf{y}\| \leq R_p(h)} K\left(\frac{u_p(\mathbf{y})}{h}\right) \xi(\mathbf{y})\sqrt{g(\mathbf{y})}dy^1 \ldots dy^n , \tag{15}$$

where $g$ denotes the determinant of $\mathbf{g}$ as calculated in normal coordinates. Changing the variable of integration to $\mathbf{z} = \mathbf{y}/h$, we get,

$$\xi_h(p) - \xi(p) \int K(\|\mathbf{z}\|)d^n\mathbf{z} =$$

$$\int_{\|\mathbf{z}\| \le R_p(h)/h} K\left(\frac{u_p(\mathbf{z}h)}{h}\right) \xi(\mathbf{z}h)\sqrt{g(\mathbf{z}h)}d^n\mathbf{z} - \xi(\mathbf{0}) \int_{\|\mathbf{z}\| \le 1} K(\|\mathbf{z}\|)d^n\mathbf{z}$$

$$= \int_{\|\mathbf{z}\| \le 1} K\left(\frac{u_p(\mathbf{z}h)}{h}\right) \xi(\mathbf{z}h)\left(\sqrt{g(\mathbf{z}h)} - 1\right)d^n\mathbf{z} +$$

$$\int_{\|\mathbf{z}\| \le 1} \xi(\mathbf{z}h)\left(K\left(\frac{u_p(\mathbf{z}h)}{h}\right) - K(\|\mathbf{z}\|)\right)d^n\mathbf{z} +$$

$$\int_{\|\mathbf{z}\| \le 1} K(\|\mathbf{z}\|)\left(\xi(\mathbf{z}h) - \xi(\mathbf{0})\right)d^n\mathbf{z} +$$

$$\int_{1 < \|\mathbf{z}\| \le R_p(h)/h} K\left(\frac{u_p(\mathbf{z}h)}{h}\right) \xi(\mathbf{z}h)\sqrt{g(\mathbf{z}h)}d^n\mathbf{z} \,.$$

Thus,

$$\left|\xi_h(p) - \xi(p) \int K\left(\|\mathbf{z}\|\right)d^n\mathbf{z}\right| \le \tag{16}$$

$$\sup_{t \in \mathbb{R}} K(t) \cdot \sup_{\|\mathbf{z}\| \le 1} |\xi(\mathbf{z}h)| \cdot \sup_{\|\mathbf{z}\| \le 1} \left|\sqrt{g(\mathbf{z}h)} - 1\right| \cdot \int_{\|\mathbf{z}\| \le 1} d^n\mathbf{z} + \tag{17}$$

$$\sup_{\|\mathbf{z}\| \le 1} |\xi(\mathbf{z}h)| \cdot \sup_{\|\mathbf{z}\| \le 1} \left|K(\frac{u_p(\mathbf{z}h)}{h}) - K(\|\mathbf{z}\|)\right| \cdot \int_{\|\mathbf{z}\| \le 1} d^n\mathbf{z} + \tag{18}$$

$$\left|\int_{\|\mathbf{z}\| \le 1} K(\|\mathbf{z}\|)(\xi(\mathbf{z}h) - \xi(\mathbf{0}))d^n\mathbf{z}\right| + \tag{19}$$

$$\sup_{t \in \mathbb{R}} K(t) \cdot \sup_{1 < \|\mathbf{z}\| \le R_p(h)/h} \sqrt{g(\mathbf{z}h)} \cdot \sup_{1 < \|\mathbf{z}\| \le R_p(h)/h} |\xi(\mathbf{z}h)| \cdot \int_{1 < \|\mathbf{z}\| \le R_p(h)/h} d^n\mathbf{z} \,. \tag{20}$$

Letting $h \to 0$, the terms (17)-(20) approach zero at the following rates:

(17): $K(t)$ is bounded and $\xi(\mathbf{y})$ is continuous at $\mathbf{y} = \mathbf{0}$, so the first two terms can be bounded by constants as $h \to 0$. In normal coordinates $\mathbf{y}$, $g_{ij}(\mathbf{y}) = \delta_{ij} + O(\|\mathbf{y}\|^2)$ as $\|\mathbf{y}\| \to 0$, so, $\sup_{\|\mathbf{z}\| \le 1} \left|\sqrt{g(\mathbf{z}h)} - 1\right| = O(h^2)$ as $h \to 0$.

(18): Since $K$ is assumed to be differentiable with a bounded derivative in $[0,1)$, we get $K(b) - K(a) = O(b-a)$ as $b \to a$. By Lemma 3.1 we have $\frac{u_p(\mathbf{z}h)}{h} - \|\mathbf{z}\| = O(h^2)$ as $h \to 0$. Thus, $K\left(\frac{u_p(\mathbf{z}h)}{h}\right) - K(\|\mathbf{z}\|) = O(h^2)$ as $h \to 0$.

(19): Since $\xi(\mathbf{y})$ is assumed to have partial derivatives up to second order in a neighborhood of $\mathbf{y}(p) = \mathbf{0}$, for $\|\mathbf{z}\| \le 1$, Taylor's theorem gives,

$$\xi(\mathbf{z}h) = \xi(\mathbf{0}) + h \sum_{i=1}^n z^i \frac{\partial \xi(\mathbf{y})}{\partial y^i}\Big|_{\mathbf{y}=0} + O(h^2) \tag{21}$$

as $h \to 0$. Since $\int_{\|\mathbf{z}\| \le 1} \mathbf{z}K(\|\mathbf{z}\|)d^n\mathbf{z} = 0$, we get $\left|\int_{\|\mathbf{z}\| \le 1} K(\|\mathbf{z}\|)(\xi(\mathbf{z}h) - \xi(\mathbf{0}))d^n\mathbf{z}\right| = O(h^2)$ as $h \to 0$.

(20): The first three terms can be bounded by constants. By Lemma 3.3, $R_p(h) = h + O(h^3)$ as $h \to 0$. A spherical shell $1 < \|\mathbf{z}\| \le 1 + \epsilon$ has volume $O(\epsilon)$ as $\epsilon \to 0^+$. Thus, the volume of $1 < \|\mathbf{z}\| \le R_p(h)/h$ is $O(R_p(h)/h - 1) = O(h^2)$ as $h \to 0$.

Thus, the sum of the terms (17-20), is $O(h^2)$ as $h \to 0$, as claimed in Theorem 3.1.

# 4  Bias, variance and mean squared error

Let $M$, $f$, $\hat{f}_m$, $K$, $p$ be as in Theorem 2.1 and assume $h_m \to 0$ as $m \to \infty$.

**Lemma 4.1.** Bias $\left[ \hat{f}_m(p) \right] = O(h_m^2)$, *as* $m \to \infty$.

*Proof.* We have $\text{Bias}[f_m(p)] = \text{Bias} \left[ \frac{1}{h_m} K \left( \frac{u_p(q)}{h} \right) \right]$, so recalling $\int_{\mathbb{R}^n} K(\|\mathbf{z}\|) d^n \mathbf{z} = 1$, the lemma follows from Theorem 3.1 with $\xi$ replaced with $f$. □

**Lemma 4.2.** *If in addition to* $h_m \to 0$, *we have* $m h_m^n \to \infty$ *as* $m \to \infty$, *then,* $\text{Var}[f_m(p)] = O\left( \frac{1}{m h_m^n} \right)$, *as* $m \to \infty$.

*Proof.*

$$\text{Var}[f_m(p)] = \frac{1}{m} \text{Var} \left[ \frac{1}{h_m^n} K \left( \frac{u_p(q)}{h_m} \right) \right] \tag{22}$$

$$\tag{23}$$

Now,

$$\text{Var} \left[ \frac{1}{h_m^n} K \left( \frac{u_p(q)}{h_m} \right) \right] = \text{E} \left[ \frac{1}{h_m^{2n}} K^2 \left( \frac{u_p(q)}{h_m} \right) \right] - \left( \text{E} \left[ \frac{1}{h_m^n} K \left( \frac{u_p(q)}{h_m} \right) \right] \right)^2 , \tag{24}$$

and,

$$\text{E} \left[ \frac{1}{h_m^{2n}} K^2 \left( \frac{u_p(q)}{h_m} \right) \right] = \frac{1}{h_m^n} \int_M f(q) \frac{1}{h_m^n} K^2 \left( \frac{u_p(q)}{h_m} \right) dV(q) . \tag{25}$$

By Theorem 3.1, the integral in (25) converges to $f(p) \int K^2(\|\mathbf{z}\|) d^n \mathbf{z}$, so, the right hand side of (25) is $O\left( \frac{1}{h_m^n} \right)$ as $m \to \infty$. By Lemma 4.1 we have, $\left( \text{E} \left[ \frac{1}{h_m^n} K \left( \frac{u_p(q)}{h_m} \right) \right] \right)^2 \to f^2(p)$. Thus, $\text{Var}[\hat{f}_m(p)] = O\left( \frac{1}{m h_m^n} \right)$ as $m \to \infty$. □

**Proof of Theorem 2.1**  Finally, since MSE $\left[ \hat{f}_m(p) \right] = \text{Bias}^2[\hat{f}_m(p)] + \text{Var}[\hat{f}_m(p)]$, the theorem follows from Lemma 4.1 and 4.2.

# 5  Experiments and discussion

We have empirically tested the estimator (3) on two datasets: A unit normal distribution mapped onto a piece of a spiral in the plane, so that $n = 1$ and $N = 2$, and a uniform distribution on the unit disc $x^2 + y^2 \leq 1$ mapped onto the unit hemisphere by $(x, y) \to (x, y, 1 - \sqrt{x^2 + y^2})$, so that $n = 2$ and $N = 3$. We picked the bandwidths to be proportional to $m^{-1/(n+4)}$ where $m$ is the number of data points. We performed live-one-out estimates of the density on the data points, and obtained the MSE for a range of $m$s. See Figure 5.

# 6  Conclusion and future work

We have proposed a small modification of the usual KDE in order to estimate the density of data that lives on an $n$-dimensional submanifold of $\mathbb{R}^N$, and proved that the rate of convergence of the estimator is determined by the intrinsic dimension $n$. This shows that the curse of dimensionality in KDE can be overcome for data with low intrinsic dimension. Our method assumes that the intrinsic dimensionality $n$ is given, so it has to be supplemented with an estimator of the dimension. We have assumed various smoothness properties for the submanifold $M$, the density $f$, and the kernel $K$. We find it likely that our estimator or slight modifications of it will be consistent under weaker requirements. Such a relaxation of requirements would have practical consequences, since it is unlikely that a generic data set lives on a smooth Riemannian manifold.

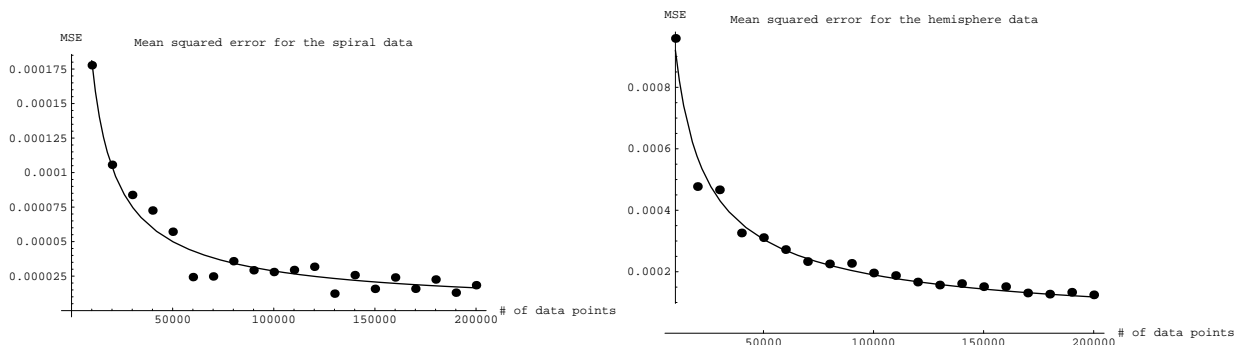

Figure 1: Mean squared error as a function of the number of data points for the spiral data (left) and the hemisphere data. In each case, we fit a curve of the form $MSE(m) = am^b$, which gave $b = -0.80$ for the spiral and $b = -0.69$ for the hemisphere. Theorem 2.1 bounds the MSE by $Cm^{-4/(n+4)}$, which gives the exponent as $-0.80$ for the spiral and $-0.67$ for the hemisphere.

## Footnotes

[1]The metric tensor can be thought of as giving the "infinitesimal distance" $ds$ between two points whose coordinates differ by the infinitesimal amounts $(dy^1, \ldots, dy^N)$ as $ds^2 = \sum_{ij} g_{ij} dy^i dy^j$.

[2]The injectivity radius $r_{inj}$ of a Riemannian manifold is a distance such that all geodesic pieces (i.e., curves with zero intrinsic acceleration) of length less than $r_{inj}$ minimize the length between their endpoints. On a complete Riemannian manifold, there exists a distance-minimizing geodesic between any given pair of points, however, an arbitrary geodesic need not be distance minimizing. For example, any two non-antipodal points on the sphere can be connected with two geodesics with different lengths, namely, the two pieces of the great circle passing throught the points. For a detailed discussion of these issues, see, e.g., [1].

[3]Primes denote differentiation with respect to s.

[4]Normal coordinates at a point $p$ in a Riemannian manifold are a close approximation to Cartesian coordinates, in the sense that the components of the metric have vanishing first derivatives at $p$, and $g_{ij}(p) = \delta_{ij}$ [1]. Normal coordinates can be defined in a "geodesic ball" of radius less than $r_{inj}$.

# References

[1] M. Berger and N. Hitchin. A panoramic view of Riemannian geometry. *The Mathematical Intelligencer*, 28(2):73–74, 2006.

[2] A. Beygelzimer, S. Kakade, and J. Langford. Cover trees for nearest neighbor. In *Proceedings of the 23rd international conference on Machine learning*, pages 97–104. ACM New York, NY, USA, 2006.

[3] T. Cacoullos. Estimation of a multivariate density. *Annals of the Institute of Statistical Mathematics*, 18(1):179–189, 1966.

[4] H. Hendriks. Nonparametric estimation of a probability density on a Riemannian manifold using Fourier expansions. *The Annals of Statistics*, 18(2):832–849, 1990.

[5] J. Jost. *Riemannian geometry and geometric analysis*. Springer, 2008.

[6] F. Korn, B. Pagel, and C. Faloutsos. On dimensionality and self-similarity . *IEEE Transactions on Knowledge and Data Engineering*, 13(1):96–111, 2001.

[7] J. Lee. *Riemannian manifolds: an introduction to curvature*. Springer Verlag, 1997.

[8] E. Parzen. On estimation of a probability density function and mode. *The Annals of Mathematical Statistics*, pages 1065–1076, 1962.

[9] B. Pelletier. Kernel density estimation on Riemannian manifolds. *Statistics and Probability Letters*, 73(3):297–304, 2005.

[10] X. Pennec. Probabilities and statistics on Riemannian manifolds: Basic tools for geometric measurements. In *IEEE Workshop on Nonlinear Signal and Image Processing*, volume 4. Citeseer, 1999.

[11] X. Pennec. Intrinsic statistics on Riemannian manifolds: Basic tools for geometric measurements. *Journal of Mathematical Imaging and Vision*, 25(1):127–154, 2006.

[12] B. Silverman. *Density estimation for statistics and data analysis*. Chapman & Hall/CRC, 1986.

[13] P. Vincent and Y. Bengio. Manifold Parzen Windows. *Advances in Neural Information Processing Systems*, pages 849–856, 2003.

